# Convergence and No-Regret in Multiagent Learning

**Michael Bowling**
Department of Computing Science
University of Alberta
Edmonton, Alberta
Canada T6G 2E8
bowling@cs.ualberta.ca

## Abstract

Learning in a multiagent system is a challenging problem due to two key factors. First, if other agents are simultaneously learning then the environment is no longer stationary, thus undermining convergence guarantees. Second, learning is often susceptible to deception, where the other agents may be able to exploit a learner's particular dynamics. In the worst case, this could result in poorer performance than if the agent was not learning at all. These challenges are identifiable in the two most common evaluation criteria for multiagent learning algorithms: convergence and regret. Algorithms focusing on convergence or regret in isolation are numerous. In this paper, we seek to address both criteria in a single algorithm by introducing GIGA-WoLF, a learning algorithm for normal-form games. We prove the algorithm guarantees at most zero average regret, while demonstrating the algorithm converges in many situations of self-play. We prove convergence in a limited setting and give empirical results in a wider variety of situations. These results also suggest a third new learning criterion combining convergence and regret, which we call negative non-convergence regret (NNR).

## 1 Introduction

Learning to select actions to achieve goals in a multiagent setting requires overcoming a number of key challenges. One of these challenges is the loss of the stationarity assumption when multiple agents are learning simultaneously. Another challenge is guaranteeing that the learner cannot be deceptively exploited by another agent. Both of these challenges distinguish the multiagent learning problem from traditional single-agent learning, and have been gaining recent attention as multiagent applications continue to proliferate.

In single-agent learning tasks, it is reasonable to assume that the same action from the same state will result in the same distribution over outcomes, both rewards and next states. In other words, the environment is stationary. In a multiagent task with other learning agents, the outcomes of an agent's action will vary with the changing policies of the other agents. Since most of the convergence results in reinforcement learning depend upon the environment being stationary, convergence is often difficult to obtain in multiagent settings.

The desirability of convergence has been recently contested. We offer some brief insight into this debate in the introduction of the extended version of this paper [1].

Equilibrium learners [2, 3, 4] are one method of handling the loss of stationarity. These algorithms learn joint-action values, which *are* stationary, and in certain circumstances guarantee these values converge to Nash (or correlated) equilibrium values. Using these values, the player's strategy corresponds to the player's component of some Nash (or correlated) equilibrium. This convergence of strategies is guaranteed nearly independently of the actions selected by the other agents, including when other agents play suboptimal responses. Equilibrium learners, therefore, can fail to learn best-response policies even against simple non-learning opponents.[1] Best-response learners [5, 6, 7] are an alternative approach that has sought to learn best-responses, but still considering whether the resulting algorithm converges in some form. These approaches usually examine convergence in self-play, and have included both theoretical and experimental results.

The second challenge is the avoidance of exploitation. Since learning strategies dynamically change their action selection over time, it is important to know that the change cannot be exploited by a clever opponent. A deceptive strategy may "lure" a dynamic strategy away from a safe choice in order to switch to a strategy where the learner receives much lower reward. For example, Chang and Kaelbling [8] demonstrated that the best-response learner PHC [7] could be exploited by a particular dynamic strategy. One method of measuring whether an algorithm can be exploited is the notion of regret. Regret has been explored both in game theory [9] and computer science [10, 11]. Regret measures how much worse an algorithm performs compared to the best static strategy, with the goal to guarantee at least zero average regret, *no-regret*, in the limit.

These two challenges result in two completely different criteria for evaluation: convergence and no-regret. In addition, they have almost exclusively been explored in isolation. For example, equilibrium learners can have arbitrarily large average regret. On the other hand, no-regret learners' strategies rarely converge in self-play [12] in even the simplest of games.[2] In this paper, we seek to explore these two criteria in a single algorithm for learning in normal-form games. In Section 2 we present a more formal description of the problem and the two criteria. We also examine key related work in applying gradient ascent algorithms to this learning problem. In Section 3 we introduce GIGA-WoLF, an algorithm with both regret and convergence properties. The algorithm is followed by theoretical and experimental analyses in Sections 4 and 5, respectively, before concluding.

## 2 Online Learning in Games

A game in normal form is a tuple, $(n, \mathcal{A}_{1 \ldots n}, R_{1 \ldots n})$, where $n$ is the number of players in the game, $\mathcal{A}_i$ is a set of actions available to player $i$ ($\mathcal{A} = \mathcal{A}_1 \times \ldots \times \mathcal{A}_n$), and $R_i : \mathcal{A} \to \Re$ is a mapping from joint actions to player $i$'s reward. The problem of learning in a normal-form game is one of repeatedly selecting an action and receiving a reward, with a goal of maximizing average reward against an unknown opponent. If there are two players then it is convenient to write a player's reward function as a $|\mathcal{A}_1| \times |\mathcal{A}_2|$ matrix. Three example normal-form games are shown in Table 1.

Unless stated otherwise we will assume the learning algorithm is player one. In the context of a particular learning algorithm and a particular opponent, let $r_t \in \Re^{|\mathcal{A}_1|}$ be the vector of actual rewards that player one would receive at time $t$ for each of its actions. Let $x_t \in$

Table 1: Examples of games in normal-form.

$$R_1 = \begin{matrix} & \mathbf{H} \quad \mathbf{T} \\ \mathbf{H} \\ \mathbf{T} \end{matrix} \begin{pmatrix} 1 & -1 \\ -1 & 1 \end{pmatrix} \qquad R_1 = \begin{matrix} & \mathbf{A} \ \mathbf{B} \\ \mathbf{A} \\ \mathbf{B} \end{matrix} \begin{pmatrix} 0 & 3 \\ 1 & 2 \end{pmatrix} \qquad R_1 = \begin{matrix} & \mathbf{R} \quad \mathbf{P} \quad \mathbf{S} \\ \mathbf{R} \\ \mathbf{P} \\ \mathbf{S} \end{matrix} \begin{pmatrix} 0 & -1 & 1 \\ 1 & 0 & -1 \\ -1 & 1 & 0 \end{pmatrix}$$

$$R_2 = -R_1 \qquad\qquad R_2 = \begin{matrix} \mathbf{A} \\ \mathbf{B} \end{matrix} \begin{pmatrix} 3 & 2 \\ 0 & 1 \end{pmatrix}$$

$$\qquad\qquad\qquad\qquad\qquad\qquad R_2 = -R_1$$

(a) Matching Pennies  (b) Tricky Game  (c) Rock–Paper–Scissors

$PD(\mathcal{A}_1)$ be the algorithm's strategy at time $t$, selected from probability distributions over actions. So, player one's expected payoff at time $t$ is $(r_t \cdot x_t)$. Let $1_a$ be the probability distribution that assigns probability 1 to action $a \in \mathcal{A}_1$. Lastly, we will assume the reward for any action is bounded by $r_{\max}$ and therefore $||r_t||^2 \le |\mathcal{A}_1| r_{\max}^2$.

## 2.1 Evaluation Criteria

One common evaluation criterion for learning in normal-form games is convergence. There are a number of different forms of convergence that have been examined in the literature. These include, roughly increasing in strength: average reward (i.e., $\sum (r_t \cdot x_t)/T$), empirical distribution of actions (i.e., $\sum x_t/T$), expected reward (i.e., $(r_t \cdot x_t)$), and strategies (i.e., $x_t$). We focus in this paper on convergence of strategies as this implies the other three forms of convergence as well. In particular, we will say an algorithm converges against a particular opponent if and only if $\lim_{t \to \infty} x_t = x_*$.

The second common evaluation criterion is regret. Total regret[3] is the difference between the maximum total reward of any static strategy given the past history of play and the algorithm's total reward.

$$\mathcal{R}_T \equiv \max_{a \in \mathcal{A}_1} \sum_{t=1}^{T} ((r_t \cdot 1_a) - (r_t \cdot x_t))$$

Average regret is just the asymptotic average of total regret, $\lim_{T \to \infty} \mathcal{R}_T/T$. An algorithm has no-regret if and only if the average regret is less than or equal to zero against all opponent strategies. The no-regret property makes a strong claim about the performance of the algorithm: the algorithm's expected average reward is at least as large as the expected average award any static strategy could have achieved. In other words, the algorithm is performing at least as well as any static strategy.

## 2.2 Gradient Ascent Learning

Gradient ascent is a simple and common technique for finding parameters that optimize a target function. In the case of learning in games, the parameters represent the player's strategy, and the target function is expected reward. We will examine three recent results evaluating gradient ascent learning algorithms in normal-form games.

Singh, Kearns, and Mansour [6] analyzed gradient ascent (IGA) in two-player, two-action games, e.g., Table 1(a) and (b). They examined the resulting strategy trajectories and payoffs in self-play, demonstrating that strategies do not always converge to a Nash equilibrium, depending on the game. They proved, instead, that average payoffs converge (a

weaker form of convergence) to the payoffs of the equilibrium. WoLF-IGA [7] extended this work to the stronger form of convergence, namely convergence of strategies, through the use of a variable learning rate. Using the WoLF ("Win or Learn Fast") principle, the algorithm would choose a larger step size when the current strategy had less expected payoff than the equilibrium strategy. This results in strategies converging to the Nash equilibrium in a variety of games including all two-player, two-action games.[4] Zinkevich [11] looked at gradient ascent using the evaluation criterion of regret. He first extended IGA beyond two-player, two-action games. His algorithm, GIGA (Generalized Infinitesimal Gradient Ascent), updates strategies using an unconstrained gradient, and then projects the resulting strategy vector back into the simplex of legal probability distributions,

$$x_{t+1} = P(x_t + \eta_t r_t) \qquad \text{where} \qquad P(x) = \operatorname*{argmin}_{x' \in PD(\mathcal{A}_1)} ||x - x'||, \qquad (1)$$

$\eta_t$ is the stepsize at time $t$, and $||\cdot||$ is the standard L2 norm. He proved GIGA's total regret is bounded by,

$$\mathcal{R}_T \quad \leq \quad \sqrt{T} + |A| r_{\max}^2 (\sqrt{T} - 1/2). \qquad (2)$$

Since GIGA is identical to IGA in two-player, two-action games, we also have that GIGA achieves the weak form of convergence in this subclass of games. It is also true, though, that GIGA's strategies do not converge in self-play even in simple games like matching pennies.

In the next section, we present an algorithm that simultaneously achieves GIGA's no-regret result and part of WoLF-IGA's convergence result. We first present the algorithm and then analyze these criteria both theoretically and experimentally.

## 3 GIGA-WoLF

GIGA-WoLF is a gradient based learning algorithm that internally keeps track of two different gradient-updated strategies, $x_t$ and $z_t$. The algorithm chooses actions according to the distribution $x_t$, but updates both $x_t$ and $z_t$ after each iteration. The update rules consist of three steps.

$$
\begin{aligned}
(1) \quad \hat{x}_{t+1} &= P(x_t + \eta_t r_t) \\
(2) \quad z_{t+1} &= P(z_t + \eta_t r_t / 3) \\
\delta_{t+1} &= \min\left(1, \frac{||z_{t+1} - z_t||}{||z_{t+1} - \hat{x}_{t+1}||}\right) \\
(3) \quad x_{t+1} &= \hat{x}_{t+1} + \delta_{t+1}(z_{t+1} - \hat{x}_{t+1})
\end{aligned}
$$

Step (1) updates $x_t$ according to GIGA's standard gradient update and stores the result as $\hat{x}_{t+1}$. Step (2) updates $z_t$ in the same manner, but with a smaller step-size. Step (3) makes a final adjustment on $x_{t+1}$ by moving it toward $z_{t+1}$. The magnitude of this adjustment is limited by the change in $z_t$ that occurred in step (2).

A key factor in understanding this algorithm is the observance that a strategy $a$ receives higher reward than a strategy $b$ if and only if the gradient at $a$ is in the direction of $b$ (i.e., $r_t \cdot (b - a) > 0$). Therefore, the step (3) adjustment is in the direction of the gradient if and only if $z_t$ received higher reward than $x_t$. Notice also that, as long as $x_t$ is not near the boundary, the change due to step (3) is of lower magnitude than the change due

to step (1). Hence, the combination of steps (1) and (3) result in a change with two key properties. First, the change is in the direction of positive gradient. Second, the magnitude of the change is larger when $z_t$ received higher reward than $x_t$. So, we can interpret the update rule as a variation on the WoLF principle of "win or learn fast", i.e., the algorithm is *learning faster* if and only if its strategy $x$ is *losing* to strategy $z$. GIGA-WoLF is a major improvement on the original presentation of WoLF-IGA, where winning was determined by comparison with an equilibrium strategy that was assumed to be given. Not only is less knowledge required, but the use of a GIGA-updated strategy to determine winning will allow us to prove guarantees on the algorithm's regret.

In the next section we present a theoretical examination of GIGA-WoLF's regret in $n$-player, $n$-action games, along with a limited guarantee of convergence in two-player, two-action games. In Section 5, we give experimental results of learning using GIGA-WoLF, demonstrating that convergence extends beyond the theoretical analysis presented.

## 4   Theoretical Analysis

We begin by examining GIGA-WoLF's regret against an unknown opponent strategy. We will prove the following bound on average regret.

**Theorem 1**  *If $\eta_t = 1/\sqrt{t}$, the regret of GIGA-WoLF is,*

$$\mathcal{R}_T \leq 2\sqrt{T} + |A|r_{\max}^2(2\sqrt{T} - 1).$$

*Therefore, $\lim_{T \to \infty} \mathcal{R}_T/T \leq 0$, hence GIGA-WoLF has no-regret.*

**Proof.**   We begin with a brief overview of the proof. We will find a bound on the regret of the strategy $x_t$ with respect to the dynamic strategy $z_t$. Since $z_t$ is unmodified GIGA, we already have a bound on the regret of $z_t$ with respect to any static strategy. Hence, we can bound the regret of $x_t$ with respect to any static strategy.

We start by examining the regret of $x_t$ with respect to $z_t$ using a similar analysis as used by Zinkevich [11]. Let $\rho_t^{x \to z}$ refer to the difference in expected payoff between $z_t$ and $x_t$ at time $t$, and $\mathcal{R}_T^{x \to z}$ be the sum of these differences, i.e., the total regret of $x_t$ with respect to $z_t$,

$$\rho_t^{x \to z} = r_t \cdot (z_t - x_t) \qquad \mathcal{R}_T^{x \to z} \equiv \sum_{t=1}^{T} \rho_t^{x \to z}.$$

We will use the following potential function, $\Phi_t \equiv (x_t - z_t)^2/2\eta_t$. We can examine how this potential changes with each step of the update. $\Delta\Phi_t^1$, $\Delta\Phi_t^2$, and $\Delta\Phi_t^3$ refers to the change in potential caused by steps (1), (2), and (3), respectively. $\Delta\Phi_t^4$ refers to the change in potential caused by the learning rate change from $\eta_{t-1}$ to $\eta_t$. This gives us the following equations for the potential change.

$$
\begin{aligned}
\Delta\Phi_{t+1}^1 &= 1/2\eta_t((\hat{x}_{t+1} - z_t)^2 - (x_t - z_t)^2) \\
\Delta\Phi_{t+1}^2 &= 1/2\eta_t((\hat{x}_{t+1} - z_{t+1})^2 - (\hat{x}_{t+1} - z_t)^2) \\
\Delta\Phi_{t+1}^3 &= 1/2\eta_t((x_{t+1} - z_{t+1})^2 - (\hat{x}_{t+1} - z_{t+1})^2) \\
\Delta\Phi_{t+1}^4 &= (1/2\eta_{t+1} - 1/2\eta_t)(x_{t+1} - z_{t+1})^2 \\
\Delta\Phi_{t+1} &= \Delta\Phi_{t+1}^1 + \Delta\Phi_{t+1}^2 + \Delta\Phi_{t+1}^3 + \Delta\Phi_{t+1}^4
\end{aligned}
$$

Notice that if $\delta_{t+1} = 1$ then $x_{t+1} = z_{t+1}$. Hence $\Phi_{t+1} = 0$, and $\Delta\Phi_{t+1}^2 + \Delta\Phi_{t+1}^3 \leq 0$. If $\delta_{t+1} < 1$, then $||x_{t+1} - \hat{x}_{t+1}|| = ||z_{t+1} - z_t||$. Notice also that in this case $x_{t+1}$ is

co-linear and between $\hat{x}_{t+1}$ and $z_{t+1}$. So,

$$
\begin{aligned}
||\hat{x}_{t+1} - z_{t+1}|| &= ||\hat{x}_{t+1} - x_{t+1}|| + ||x_{t+1} - z_{t+1}|| \\
&= ||z_{t+1} - z_t|| + ||x_{t+1} - z_{t+1}||
\end{aligned}
$$

We can bound the left with the triangle inequality,

$$
\begin{aligned}
||\hat{x}_{t+1} - z_{t+1}|| &\leq ||\hat{x}_{t+1} - z_t|| + ||z_t - z_{t+1}|| \\
||x_{t+1} - z_{t+1}|| &\leq ||\hat{x}_{t+1} - z_t||.
\end{aligned}
$$

So regardless of $\delta_{t+1}$, $\Delta\Phi_{t+1}^2 + \Delta\Phi_{t+1}^3 < 0$. Hence, $\Delta\Phi_{t+1} \leq \Delta\Phi_{t+1}^1 + \Delta\Phi_{t+1}^4$.

We will now use this bound on the change in the potential to bound the regret of $x_t$ with respect to $z_t$. We know from Zinkevich that,

$$
(\hat{x}_{t+1} - z_t)^2 - (x_t - z_t)^2 \leq 2\eta_t r_t \cdot (z_t - x_t) + \eta_t^2 r_t^2.
$$

Therefore,

$$
\begin{aligned}
\rho_t^{x \to z} &\leq -\frac{1}{2\eta_t} \left( (\hat{x}_{t+1} - z_t)^2 - (x_t - z_t)^2 - r_t^2 \right) \\
&\leq -\Delta\Phi_{t+1}^1 + 1/2\eta_t r_t^2 \quad \leq \quad -\Delta\Phi_{t+1} + \Delta\Phi_{t+1}^4 + 1/2\eta_t r_t^2.
\end{aligned}
$$

Since we assume rewards are bounded by $r_{\max}$ we can bound $r_t^2$ by $|A|r_{\max}^2$. Summing up regret and using the fact that $\eta_t = 1/\sqrt{t}$, we get the following bound.

$$
\begin{aligned}
\mathcal{R}_T^{x \to z} &\leq \sum_{t=1}^T -\Delta\Phi_t + \Delta\Phi_t^4 + \frac{\eta_t}{2}|A|r_{\max}^2 \\
&\leq (\Phi_1 - \Phi_T) + \left( \frac{1}{\eta_T} - 1 \right) + \frac{|A|r_{\max}^2}{2} \sum_{t=1}^T \eta_t \\
&\leq \sqrt{T} + |A|r_{\max}^2(\sqrt{T} - 1/2)
\end{aligned}
$$

We know that GIGA's regret with respect to any strategy is bounded by the same value (see Inequality 2). Hence,

$$
\mathcal{R}_T \leq 2\sqrt{T} + |A|r_{\max}^2(2\sqrt{T} - 1),
$$

as claimed. $\qquad\square$

The second criterion we want to consider is convergence. As with IGA, WoLF-IGA, and other algorithms, our theoretical analysis will be limited to two-player, two-action general-sum games. We further limit ourselves to the situation of GIGA-WoLF playing "against" GIGA. These restrictions are a limitation of the proof method, which uses a case-by-case analysis that is combinatorially impractical for the case of self-play. This is not necessarily a limitation on GIGA-WoLF's convergence. This theorem along with the empirical results we present later in Section 5 give a strong sense of GIGA-WoLF's convergence properties. The full proof can be found in [1].

**Theorem 2** *In a two-player, two-action repeated game, if one player follows the GIGA-WoLF algorithm and the other follows the GIGA algorithm, then their strategies will converge to a Nash equilibrium.*

## 5 Experimental Analysis

We have presented here two theoretical properties of GIGA-WoLF relating to guarantees on both regret and convergence. There have also been extensive experimental results performed on GIGA-WoLF in a variety of normal-form games [1]. We summarize the results

here. The purpose of these experiments was to demonstrate the theoretical results from the previous section as well as explore the extent to which the results (convergence, in particular) can be generalized. In that vein, we examined the same suite of normal-form games used in experiments with WoLF-PHC, the practical variant of WoLF-IGA [7].

One of the requirements of GIGA-WoLF (and GIGA) is knowledge of the entire reward vector ($r_t$), which requires knowledge of the game and observation of the opponent's action. In practical situations, one or both of these are unlikely to be available. Instead, only the reward of the selected action is likely to be observable. We have relaxed this requirement in these experiments by providing GIGA-WoLF (and GIGA) with only estimates of the gradient from stochastic approximation. In particular, after selecting action $a$ and receiving reward $\hat{r}_a$, we update the current estimate of action $a$'s component of the reward vector, $r_{t+1} = r_t + \alpha_t(\hat{r}_a - 1_a \cdot r_t)1_a$, where $\alpha_t$ is the learning rate. This is a standard method of estimation commonly used in reinforcement learning (e.g., Q-learning).

For almost all of the games explored, including two-player, two-action games as well as $n$-action zero-sum games, GIGA-WoLF strategies converged in self-play to equilibrium strategies of the game. GIGA's strategies, on the other hand, failed to converge in self-play over the same suite of games. These results are nearly identical to the PHC and WoLF-PHC experiments over the same games. A prototypical example of these results is provided in Figure 1(a) and (b), showing strategy trajectories while learning in Rock-Paper-Scissors. GIGA's strategies do not converge, while GIGA-WoLF's strategies do converge. GIGA-WoLF also played directly against GIGA in this game resulting in convergence, but with a curious twist. The resulting expected and average payoffs are shown in Figure 1(c). Since both are no-regret learners, average payoffs are guaranteed to go to zero, but the short-term payoff is highly favoring GIGA-WoLF. This result raises an interesting question about the relative short-term performance of no-regret learning algorithms, which needs to be explored further.

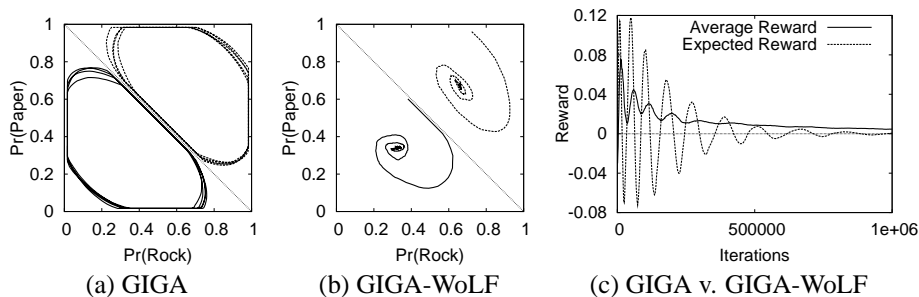

(a) GIGA      (b) GIGA-WoLF      (c) GIGA v. GIGA-WoLF

Figure 1: Trajectories of joint strategies in Rock-Paper-Scissors when both players use GIGA (a) or GIGA-WoLF (b). Also shown (c) are the expected and average payoffs of the players when GIGA and GIGA-WoLF play against each other.

GIGA-WoLF did not lead to convergence in all of the explored games. The "problematic" Shapley's game, for which many similarly convergent algorithms fail in, also resulted in non-convergence for GIGA-WoLF. On the other hand, this game has the interesting property that both players' when using GIGA-WoLF (or GIGA) actually achieve *negative* regret. In other words, the algorithms are outperforming any static strategy to which they could converge upon. This suggests a new desirable property for future multiagent (or online) learning algorithms, negative non-convergence regret (NNR). An algorithm has NNR, if it satisfies the no-regret property and either (i) achieves negative regret or (ii) its strategy converges. This property combines the criteria of regret and convergence, and GIGA-WoLF is a natural candidate for achieving this compelling result.

# 6 Conclusion

We introduced GIGA-WoLF, a new gradient-based algorithm, that we believe is the first to simultaneously address two criteria: no-regret and convergence. We proved GIGA-WoLF has no-regret. We also proved that in a small class of normal-form games, GIGA-WoLF's strategy when played against GIGA will converge to a Nash equilibrium. We summarized experimental results of GIGA-WoLF playing in a variety of zero-sum and general-sum games. These experiments verified our theoretical results and exposed two interesting phenomenon that deserve further study: short-term performance of no-regret learners and the new desiderata of negative non-convergence regret. We expect GIGA-WoLF and these results to be the foundation for further understanding of the connections between the regret and convergence criteria.

## Footnotes

[1]This work is not restricted to zero-sum games and our use of the word "opponent" refers simply to other players in the game.

[2]A notable exception is Hart and Mas-Colell's algorithm that guarantees the empirical distribution of play converges to that of a correlated equilibrium. Neither strategies nor expected values necessarily converge, though.

[3]Our analysis focuses on expectations of regret (total and average), similar to [10, 11]. Although note that for any self-oblivious behavior, including GIGA-WoLF, average regret of at most zero on expectation implies universal consistency, i.e., regret of at most $\epsilon$ with high probability [11].

[4]WoLF-IGA may, in fact, be a limited variant of the extragradient method [13] for variational inequality problems. The extragradient algorithm is guaranteed to converge to a Nash equilibrium in self-play for all zero-sum games. Like WoLF-IGA, though, it does not have any known regret guarantees, but more importantly requires the other players' payoffs to be known.

# References

[1] Michael Bowling. Convergence and no-regret in multiagent learning. Technical Report TR04-11, Department of Computing Science, University of Alberta, 2004.

[2] Michael L. Littman. Markov games as a framework for multi-agent reinforcement learning. In *Proceedings of the Eleventh International Conference on Machine Learning*, pages 157–163, 1994.

[3] Junling Hu and Michael P. Wellman. Multiagent reinforcement learning: Theoretical framework and an algorithm. In *Proceedings of the Fifteenth International Conference on Machine Learning*, pages 242–250, 1998.

[4] Amy Greenwald and Keith Hall. Correlated Q-learning. In *Proceedings of the AAAI Spring Symposium Workshop on Collaborative Learning Agents*, 2002.

[5] Caroline Claus and Craig Boutilier. The dynamics of reinforcement learning in cooperative multiagent systems. In *Proceedings of the Fifteenth National Conference on Artificial Intelligence*, pages 746–752, 1998.

[6] Satinder Singh, Michael Kearns, and Yishay Mansour. Nash convergence of gradient dynamics in general-sum games. In *Proceedings of the Sixteenth Conference on Uncertainty in Artificial Intelligence*, pages 541–548, 2000.

[7] Michael Bowling and Manuela Veloso. Multiagent learning using a variable learning rate. *Artificial Intelligence*, 136:215–250, 2002.

[8] Yu-Han Chang and Leslie Pack Kaelbling. Playing is believing: the role of beliefs in multi-agent learning. In *Advances in Neural Information Processing Systems 14*, 2001.

[9] Sergiu Hart and Andreu Mas-Colell. A simple adaptive procedure leading to correlated equilibrium. *Econometrica*, 68:1127–1150, 2000.

[10] Peter Auer, Nicolò Cesa-Bianchi, Yoav Freund, and Robert E. Schapire. Gambling in a rigged casino: The adversarial multi-arm bandit problem. In *36th Annual Symposium on Foundations of Computer Science*, pages 322–331, 1995.

[11] Martin Zinkevich. Online convex programming and generalized infinitesimal gradient ascent. In *Proceedings of the Twentieth International Conference on Machine Learning*, pages 928–925, 2003.

[12] Amir Jafari, Amy Greenwald, David Gondek, and Gunes Ercal. On no-regret learning, fictitious play, and nash equilibrium. In *Proceedings of the Eighteenth International Conference on Machine Learning*, pages 226–223, 2001.

[13] G. M. Korpelevich. The extragradient method for finding saddle points and other problems. *Matecon*, 12:747–756, 1976.
